# A Maximum-Likelihood Approach to Modeling Multisensory Enhancement

**Hans Colonius***
Institut für Kognitionsforschung
Carl von Ossietzky Universität
Oldenburg, D-26111
*hans.colonius@uni-oldenburg.de*

**Adele Diederich**
School of Social Sciences
International University Bremen
Bremen, D-28725
*a.diederich@iu-bremen.de*

## Abstract

Multisensory response enhancement (MRE) is the augmentation of the response of a neuron to sensory input of one modality by simultaneous input from another modality. The maximum likelihood (ML) model presented here modifies the Bayesian model for MRE (Anastasio et al.) by incorporating a decision strategy to maximize the number of correct decisions. Thus the ML model can also deal with the important tasks of stimulus discrimination and identification in the presence of incongruent visual and auditory cues. It accounts for the inverse effectiveness observed in neurophysiological recording data, and it predicts a functional relation between uni- and bimodal levels of discriminability that is testable both in neurophysiological and behavioral experiments.

## 1 Introduction

In a typical environment stimuli occur at various positions in space and time. In order to produce a coherent assessment of the external world an individual must constantly discriminate between signals relevant for action planning (targets) and signals that need no immediate response (distractors). Separate sensory channels process stimuli by modality, but an individual must determine which stimuli are related to one another, i.e., it is must construct a perceptual event by integrating information from several modalities. For example, stimuli that occur at the same time and space are likely to be interrelated by a common cause. However, if the visual and auditory cues are incongruent, e.g., when dubbing one syllable onto a movie showing a person mouthing a different syllable, listeners typically report hearing a third syllable that represents a combination of what was seen and heard (McGurk effect, cf. [1]). This indicates that cross-modal synthesis is particularly important for stimulus identification and discrimination, not only for detection.

Evidence for multisensory integration at the neural level has been well documented in a series of studies in the mammalian midbrain by Stein, Meredith and Wallace (e.g., [2]; for a review, see [3]). The deep layers of the superior colliculus (DSC)

integrate multisensory input and trigger orienting responses toward salient targets. Individual DSC neurons can receive inputs from multiple sensory modalities (visual, auditory, and somatosensory), there is considerable overlap between the receptive fields of these individual multisensory neurons, and the number of neural impulses evoked depends on the spatial and temporal relationships of the multisensory stimuli.

*Multisensory response enhancement* refers to the augmentation of the response of a DSC neuron to a multisensory stimulus compared to the response elicited by the most effective single modality stimulus. A quantitative measure of the percent enhancement is

$$MRE = \frac{CM - SM_{max}}{SM_{max}} \times 100,$$ (1)

where $CM$ is the mean number of impulses evoked by the combined-modality stimulus in a given time interval, and $SM_{max}$ refers to the response of the most effective single-modality stimulus (cf. [4]). Response enhancement in the DSC neurons can be quite impressive, with values of $MRE$ sometimes reaching values above 1000. Typically, this enhancement is most dramatic when the unimodal stimuli are weak and/or ambiguous, a principle referred to in [4] as "inverse effectiveness".

Since DSC neurons play an important role in orienting responses (like eye and head movements) to exogenous target stimuli, it is not surprising that multisensory enhancement is also observed at the behavioral level in terms of, for example, a lowering of detection thresholds or a speed-up of (saccadic) reaction time (e.g., [5], [6], [7]; see [8] for a review). Inverse effectiveness makes intuitive sense in the behavioral situation: the detection probability for a weak or ambiguous stimulus gains more from response enhancement by multisensory integration than a high-intensity stimulus that is easily detected by a single modality alone.

A model of the functional significance of multisensory enhancement has recently been proposed by Anastasio, Patton, and Belkacem-Boussaid [9]. They suggested that the responses of individual DSC neurons are proportional to the Bayesian probability that a target is present given their sensory inputs. Here, this Bayesian model is extended to yield a more complete account of the decision situation an organism is faced with. As noted above, in a natural environment an individual is confronted with the task of discriminating between stimuli important for survival ("targets") and stimuli that are irrelevant ("distractors"). Thus, an organism must not only keep up a high rate of detecting targets but, at the same time, must strive to minimize "false alarms" to irrelevant stimuli. An optimally adapted system will be one that maximizes the number of correct decisions. It will be shown here that this can be achieved already at the level of individual DSC neurons by appealing to a maximum-likelihood principle, without requiring any more information than is assumed in the Bayesian model.

The next section sketches the Bayesian model by Anastasio, Patton, and Belkacem-Boussaid (Bayesian model, for short), after which a maximum-likelihood model of multisensory response enhancement will be introduced.

## 2   The Bayesian Model of Multisensory Enhancement

DSC neurons receive input from the visual and auditory systems elicited by stimuli occurring within their receptive fields[1] According to the Bayesian model, these vi-

sual and auditory inputs are represented by random variables $V$ and $A$, respectively. A binary random variable $T$ indicates whether a signal is present ($T = 1$) or not ($T = 0$). The central assumption of the model is that a DSC neuron computes the Bayesian (posterior) probability that a target is present in its receptive field given its sensory input:

$$P(T = 1 \,|\, V = v, A = a) = \frac{P(V = v, A = a \,|\, T = 1)P(T = 1)}{P(V = v, A = a)}, \qquad (2)$$

where $v$ and $a$ denote specific values of the sensory input variables. Analogous expressions hold for the two unimodal situations. The response of the DSC neuron (number of spikes in a unit time interval) is postulated to be proportional to these probabilities.

In order to arrive at quantitative predictions two more specific assumptions are made:

**(1)** the distributions of V and A, given $T = 1$ or $T = 0$, are conditionally independent, i.e.,

$$P(V = v, A = a \,|\, T) = P(V = v \,|\, T)\, P(A = a \,|\, T)$$

for any $v, a$;

**(2)** the distribution of $V$, given $T = 1$ or $T = 0$, is Poisson with $\lambda_1$ or $\lambda_0$, resp., and the distribution of $A$, given $T = 1$ or $T = 0$, is Poisson with $\mu_1$ or $\mu_0$, resp.

The conditional independence assumption means that the visibility of a target indicates nothing about its audibility, and vice-versa. The choice of the Poisson distribution is seen as a reasonable first approximation that requires only one single parameter per distribution. Finally, the computation of the posterior probability that a target is present requires specification of the a-priori probability of a target, $P(T = 1)$.

The parameters $\lambda_0$ and $\mu_0$ denote the mean intensity of the visual and auditory input, resp., when no target is present (*spontaneous* input), while $\lambda_1$ and $\mu_1$ are the corresponding mean intensities when a target is present (*driven* input). By an appropriate choice of parameter values, Anastasio et al. [9] show that the Bayesian model reproduces values of multisensory response enhancement in the order of magnitude observed in neurophysiological experiments [10]. In particular, the property of inverse effectiveness, by which the enhancement is largest for combined stimuli that evoke only small unimodal responses, is reflected by the model.

## 3 The Maximum Likelihood Model of Multisensory Enhancement

### 3.1 The decision rule

The maximum likelihood model (ML model, for short) incorporates the basic decision problem an organism is faced with in a typical environment: to discriminate between relevant stimuli (targets), i.e., signals that require immediate reaction, and irrelevant stimuli (distractors), i.e., signals that can be ignored in a given situation. In the signal-detection theory framework (cf.[11]), $P(Yes \,|\, T = 1)$ denotes the probability that the organism (correctly) decides that a target is present (*hit*), while $P(Yes \,|\, T = 0)$ denotes the probability of deciding that a target is present when in

fact only a distractor is present (*false alarm*). In order to maximize the probability of a correct response,

$$P(C) = P(Yes\,|\,T=1)\,P(T=1) + [1 - P(Yes\,|\,T=0)]P(T=0), \qquad (3)$$

the following maximum likelihood decision rule must be adopted (cf. [12]) for, e.g., the unimodal visual case:

If $P(T=1\,|\,V=v) > P(T=0\,|\,V=v)$, then decide "Yes", otherwise decide "No".

The above inequality is equivalent to

$$\frac{P(T=1\,|\,V=v)}{P(T=0\,|\,V=v)} = \frac{P(T=1)}{P(T=0)}\frac{P(V=v\,|\,T=1)}{P(V=v\,|\,T=0)} > 1,$$

where the right-most ratio is a function of $V$, $L(V)$, the *likelihood ratio*. Thus, the above rule is equivalent to:

If $L(v) > \dfrac{1-p}{p}$, then decide "Yes", otherwise decide "No",

with $p = P(T=1)$.

Since $L(V)$ is a random variable, the probability to decide "Yes", given a target is present, is

$$P\left(Yes\,|\,T=1\right) \;=\; P\left(L(V) > \frac{1-p}{p}\,\Big|\,T=1\right).$$

Assuming Poisson distributions, this equals

$$P\left(\exp(\lambda_0 - \lambda_1)\left(\tfrac{\lambda_1}{\lambda_0}\right)^V > \tfrac{1-p}{p}\,\Big|\,T=1\right)$$
$$= P(V > c\,|\,T=1),$$

with

$$c = \frac{\ln\left(\frac{1-p}{p}\right) + \lambda_1 - \lambda_0}{\ln\left(\frac{\lambda_1}{\lambda_0}\right)}.$$

In analogy to the Bayesian model, the ML model postulates that the response of a DSC neuron (number of spikes in a unit time interval) to a given target is proportional to the probability to decide that a target is present computed under the optimal (maximum likelihood) strategy defined above.

## 3.2 Predictions for Hit Probabilities

In order to compare the predictions of the ML model for unimodal vs. bimodal inputs, consider the likelihood ratio for bimodal Poisson input under conditional independence:

$$L(V,A) \;=\; \frac{P(V=v, A=a\,|\,T=1)}{P(V=v, A=a\,|\,T=0)}$$
$$=\; \exp(\lambda_0 - \lambda_1)\left(\frac{\lambda_1}{\lambda_0}\right)^V \exp(\mu_0 - \mu_1)\left(\frac{\mu_1}{\mu_0}\right)^A.$$

The probability to decide "Yes" given bimodal input amounts to, after taking logarithms,

$$P\left(\ln\left(\frac{\lambda_1}{\lambda_0}\right)V + \ln\left(\frac{\mu_1}{\mu_0}\right)A > \ln\left(\frac{1-p}{p}\right) + \lambda_1 - \lambda_0 + \mu_1 - \mu_0\,\Big|\,T=1\right)$$

Table 1: Hit probabilities and MRE for different bimodal inputs

| | Mean Driven Input | | Prob (Hit) | | | |
|---|---|---|---|---|---|---|
| | $\lambda_1$ | $\mu_1$ | $V$ Driven | $A$ Driven | $VA$ Driven | MRE |
| Low | 6 | 7 | .000 | .027 | .046 | 704 |
| | 7 | 7 | .027 | .027 | .117 | 335 |
| | 8 | 8 | .112 | .112 | .341 | 204 |
| | 8 | 9 | .112 | .294 | .528 | 79 |
| | 8 | 10 | .112 | .430 | .562 | 31 |
| Medium | 12 | 12 | .652 | .652 | .872 | 33 |
| | 12 | 13 | .652 | .748 | .895 | 20 |
| High | 16 | 16 | .873 | .873 | .984 | 13 |
| | 16 | 20 | .873 | .961 | .990 | 3 |

Note: A-priori target probability is set at $p = 0.1$. Visual and auditory inputs have spontaneous means of 5 impulses per unit time. $V$ Driven ($A$ Driven, $VA$ Driven) columns refer to the hit probabilities given a unimodal visual (resp. auditory, bimodal) target. Multisensory response enhancement (last column) is computed using Eq. (1)

For $\lambda_1/\lambda_0 = \mu_1/\mu_0$ this probability is computed directly from the Poisson distribution with mean $(\lambda_1 + \mu_1)$. Otherwise, hit probabilities follow the distribution of a linear combination of two Poisson distributed variables. Table 1 presents[2] hit probabilities and multisensory response enhancement values for different levels of mean driven input. Obviously, the ML model imitates the inverse effectiveness relation: combining weak intensity unimodal stimuli leads to a much larger response enhancement than medium or high intensity stimuli.

### 3.3 Predictions for discriminability measures

The ML model allows to assess the sensitivity of an individual DSC neuron to discriminate between target and distractor signals. Intuitively, this sensitivity should be a (decreasing) function of the amount of overlap between the driven and the spontaneous likelihood (e.g., $P(V = v \mid T = 1)$ and $P(V = v \mid T = 0)$). One possible appropriate measure of sensitivity for the Poisson observer is (cf. [12])

$$D_V = \frac{\lambda_1 - \lambda_0}{(\lambda_1 \lambda_0)^{1/4}} \quad \text{and} \quad D_A = \frac{\mu_1 - \mu_0}{(\mu_1 \mu_0)^{1/4}} \tag{4}$$

for the visual and auditory unimodal inputs, resp. A natural choice for the bimodal measure of sensitivity then is

$$D_{VA} = \frac{(\lambda_1 + \mu_1) - (\mu_0 + \lambda_0)}{[(\lambda_1 + \mu_1)(\lambda_0 + \mu_0)]^{1/4}}. \tag{5}$$

Note that, unlike the hit probabilities, the relative increase in discriminability by combining two unimodal inputs does not decrease with the intensity of the driven input (see Table 2). Rather, the relation between bimodal and unimodal discriminability measures for the input values in Table 2 is approximately of Euclidean

Table 2: Discriminability measure values and % increase for different bimodal inputs

| Mean Driven Input | | Discriminability Value | | | |
| --- | --- | --- | --- | --- | --- |
| $\lambda_1$ | $\mu_1$ | $D_V$ | $D_A$ | $D_{VA}$ | % Increase |
| 7 | 7 | .82 | .82 | 1.16 | 41 |
| 8 | 8 | 1.19 | 1.19 | 1.69 | 41 |
| 8 | 10 | 1.19 | 1.88 | 2.18 | 16 |
| 12 | 12 | 2.52 | 2.52 | 3.57 | 41 |
| 16 | 16 | 3.68 | 3.68 | 5.20 | 41 |
| 16 | 20 | 3.68 | 4.74 | 5.97 | 26 |

Note: Visual and auditory inputs have spontaneous means of 5 impulses per unit time. % Increase of $D_{VA}$ over $D_V$ and $D_A$ (last column) is computed in analogy to Eq. (1)

distance form:

$$D_{VA} \approx \sqrt{D_V^2 + D_A^2}. \qquad (6)$$

For $\lambda_1 = \mu_1$ this amounts to $D_{VA} = \sqrt{2}D_V$ yielding the 41% increase in discriminability. The fact that the discriminability measures do not follow the inverse effectiveness rule should not be not surprising: whether two stimuli are easy or hard to discriminate depends on their signal-to-noise ratio, but not on the level of intensity.

## 4 Discussion and Conclusion

The maximum likelihood model of multisensory enhancement developed here assumes that the response of a DSC neuron to a target stimulus is proportional to the hit probability under a maximum likelihood decision strategy. Obviously, no claim is made here that the neuron actually performs these computations, only that its behavior can be described approximately in this way. Similar to the Bayesian model suggested by Anastasio et al. [9], the neuron's behavior is solely based on the a-priori probability of a target and the likelihood function for the different sensory inputs. The ML model predicts the inverse effectiveness observed in neurophysiological experiments. Moreover, the model allows to derive a measure of the neuron's ability to discriminate between targets and non-targets. It makes specific predictions how uni- and bimodal discriminability measures are related and, thereby, opens up further avenues for testing the model assumptions.

The ML model, like the Bayesian model, operates at the level of a single DSC neuron. However, an extension of the model to describe multisensory population responses is desirable: First, this would allow to relate the model predictions to numerous behavioral studies about multisensory effects (e.g., [13], [14]), and, second, as a recent study by Kadunce et al. [15] suggests, the effects of multisensory spatial coincidence observed in behavioral experiments may only be reconcilable with the degree of spatial resolution achievable by a population of DSC neurons with overlapping receptive fields. Moreover, this extension might also be useful to relate behavioral and single-unit recording results to recent findings on multisensory brain areas using functional imaging techniques (e.g., King and Calvert [16]).

## Acknowledgments

This research was partially supported by a grant from Deutsche Forschungs-gemeinschaft-SFB 517 *Neurokognition* to the first author.

## Footnotes

*www.uni-oldenburg.de/psychologie/hans.colonius/index.html

[1]An extension to the trimodal situation, including somatosensory input, could be easily attained in the models discussed here.

[2]For input combinations with $\lambda_1 \neq \mu_1$ hit probabilities are estimated from samples of 1,000 pseudo-random numbers.

## References

[1] McGurk, H. & MacDonald, J. (1976). Hearing lips and seeing voices. *Nature*, 264, 746-748.

[2] Wallace, M. T., Meredith, M. A., & Stein, B. E. (1993). Converging influences from visual, auditory, and somatosensory cortices onto output neurons of the superior colliculus. *Journal of Neurophysiology*, 69, 1797-1809.

[3] Stein, B. E., & Meredith, M. A. (1996). *The merging of the senses*. Cambridge, MA: MIT Press.

[4] Meredith, M. A. & Stein, B. E. (1986a). Spatial factors determine the activity of multisensory neurons in cat superior colliculus. *Brain Research, 365(2)*, 350-354.

[5] Frens, van Opstal, & van der Willigen (1995). Spatial and temporal factors determine auditory-visual interactions in human saccadic eye movements. *Perception & Psychophysics*, 57, 802-816.

[6] Colonius, H. & Arndt, P. A. (2001). A two stage-model for visual-auditory interaction in saccadic latencies. *Perception & Psychophysics, 63*, 126-147.

[7] Stein, B. E., Meredith, M. A., Huneycutt, W. S., & McDade, L. (1989). Behavioral indices of multisensory integration: Orientation to visual cues is affected by auditory stimuli. *Journal of Cognitive Neurosciences*, 1, 12-24.

[8] Welch, R. B., & Warren, D. H. (1986). Intersensory interactions. In K. R. Boff, L. Kaufman, & J. P. Thomas (eds.), *Handbook of perception and human performance, Volume I: Sensory process and perception* (pp. 25-1-25-36) New York: Wiley

[9] Anastasio,, T. J., Patton, P. E., & Belkacem-Boussaid, K. (2000). Using Bayes' rule to model multisensory enhancement in the superior colliculus. *Neural Computation*, 12, 1165-1187.

[10] Meredith, M. A. & Stein, B. E. (1986b). Visual, auditory, and somatosensory convergence on cells in superior colliculus results in multisensory integration. *Journal of Neurophysiology*, 56(3), 640-662.

[11] Green, D. M., & Swets, J. A. (1974). *Signal detection theory and psychophysics*. New York: Krieger Publ. Co.

[12] Egan, J. P. (1975). *Signal detection theory and ROC analysis*. New York: Academic Press.

[13] Craig, A., & Colquhoun, W. P. (1976). Combining evidence presented simultaneously to the eye and the ear: A comparison of some predictive models. *Perception & Psychophysics*, 19, 473-484.

[14] Stein, B. E., London, N., Wilkinson, L. K., & Price, D. D. (1996). Enhancement of perceived visual intensity by auditory stimuli: A psychophysical analysis. *Journal of Cognitive Neuroscience*, 8, 497-506.

[15] Kadunce, D. C., Vaughan, J. W., Wallace, M. T., & Stein, B. E. (2001). The influence of visual and auditory receptive field organization on multisensory integration in the superior colliculus. *Experimental Brain Research*, 139, 303-310.

[16] King, A. J., & Calvert, G. A. (2001). Multisensory integration: Perceptual grouping by eye and ear. *Current Biology*, 11, 322-325.